# Algorithmic Stability and Generalization Performance

**Olivier Bousquet**
CMAP
Ecole Polytechnique
F-91128 Palaiseau cedex
FRANCE
*bousquet@cmapx.polytechnique.fr*

**André Elisseeff**[*]
Barnhill Technologies
6709 Waters Avenue
Savannah, GA 31406
USA
*andre@barnhilltechnologies.com*

## Abstract

We present a novel way of obtaining PAC-style bounds on the generalization error of learning algorithms, explicitly using their stability properties. A *stable* learner is one for which the learned solution does not change much with small changes in the training set. The bounds we obtain do not depend on any measure of the complexity of the hypothesis space (e.g. VC dimension) but rather depend on how the learning algorithm searches this space, and can thus be applied even when the VC dimension is infinite. We demonstrate that regularization networks possess the required stability property and apply our method to obtain new bounds on their generalization performance.

## 1  Introduction

A key issue in computational learning theory is to bound the generalization error of learning algorithms. Until recently, most of the research in that area has focused on uniform a-priori bounds giving a guarantee that the difference between the training error and the test error is uniformly small for any hypothesis in a given class. These bounds are usually expressed in terms of combinatorial quantities such as VC-dimension. In the last few years, researchers have tried to use more refined quantities to either estimate the complexity of the search space (e.g. covering numbers [1]) or to use a posteriori information about the solution found by the algorithm (e.g. margin [11]). There exist other approaches such as observed VC dimension [12], but all are concerned with structural properties of the learning systems. In this paper we present a novel way of obtaining PAC bounds for specific algorithms explicitly using their stability properties. The notion of stability, introduced by Devroye and Wagner [4] in the context of classification for the analysis of the Leave-one-out error and further refined by Kearns and Ron [8] is used here in the context of regression in order to get bounds on the empirical error rather than the leave-one-out error. This method has the nice advantage of providing bounds that do

---
[*]This work was done while the author was at Laboratoire ERIC, Université Lumière Lyon 2, 5 avenue Pierre Mendès-France, F-69676 Bron cedex, FRANCE

not depend on any complexity measure of the search space (e.g. VC-dimension or covering numbers) but rather on the way the algorithm searches this space. In that respect, our approach can be related to Freund's [7] where the estimated size of the subset of the hypothesis space actually searched by the algorithm is used to bound its generalization error. However Freund's result depends on a complexity term which we do not have here since we are not looking separately at the hypotheses considered by the algorithm and their risk.

The paper is structured as follows: next section introduces the notations and the definition of stability used throughout the paper. Section 3 presents our main result as a PAC-like theorem. In Section 4 we will prove that regularization networks are stable and apply the main result to obtain bounds on their generalization ability. A discussion of the results will be presented in Section 5.

## 2 Notations and Definitions

$\mathcal{X}$ and $\mathcal{Y}$ being respectively an input and an output space, we consider a learning set $S = \{z_1 = (x_1, y_1), .., z_m = (x_m, y_m)\}$ of size $m$ in $\mathcal{Z} = \mathcal{X} \times \mathcal{Y}$ drawn i.i.d. from an unknown distribution $D$. A learning algorithm is a function $L$ from $\mathcal{Z}^m$ into $\mathcal{Y}^{\mathcal{X}}$ mapping a learning set $S$ onto a function $f_S$ from $\mathcal{X}$ to $\mathcal{Y}$. To avoid complex notations, we consider only deterministic algorithms. It is also assumed that the algorithm $A$ is symmetric with respect to $S$, *i.e.* for any permutation over the elements of $S$, $f_S$ yields the same result. Furthermore, we assume that all functions are measurable and all sets are countable which does not limit the interest of the results presented here.

The empirical error of a function $f$ measured on the training set $S$ is:

$$R_m(f) = \frac{1}{m} \sum_{i=1}^{m} c(f, z_i)$$

$c: \mathcal{Y}^{\mathcal{X}} \times \mathcal{X} \times \mathcal{Y} \to \mathbb{R}^+$ being a cost function. The risk or generalization error can be written as:

$$R(f) = \mathbf{E}_{z \sim D} [c(f, z)]$$

The study we describe here intends to bound the difference between empirical and generalization error for specific algorithms. More precisely, our goal is to bound for any $\epsilon > 0$, the term

$$\mathbf{P}_{S \sim D^m} [|R_m(f_S) - R(f_S)| > \epsilon] \tag{1}$$

Usually, learning algorithms cannot output just any function in $\mathcal{Y}^{\mathcal{X}}$ but rather pick a function $f_S$ in a set $\mathcal{F} \subsetneq \mathcal{Y}^{\mathcal{X}}$ representing the structure or the architecture or the model. Classical VC theory deals with structural properties and aims at bounding the following quantity:

$$\mathbf{P}_{S \sim D^m} \left[ \sup_{f \in \mathcal{F}} |R_m(f) - R(f)| > \epsilon \right] \tag{2}$$

This applies to any algorithm using $\mathcal{F}$ as a hypothesis space and a bound on this quantity directly implies a similar bound on (1). However, classical bounds require the VC dimension of $\mathcal{F}$ to be finite and do not use information about algorithmic properties. For a set $\mathcal{F}$, there exists many ways to search it which may yield different performance. For instance, multilayer perceptrons can be learned by a simple back-propagation algorithm or combined with a weight decay procedure. The outcome

of the algorithm belongs in both cases to the same set of functions, although their performance can be different.

VC theory was initially motivated by empirical risk minimization (ERM) in which case the uniform bounds on the quantity (2) give tight error bounds. Intuitively, the empirical risk minimization principle relies on a uniform law of large numbers. Because it is not known in advance, what will be the minimum of the empirical risk, it is necessary to study the difference between empirical and generalization error for all possible functions in $\mathcal{F}$. If, now, we do not consider this minimum, but instead, we focus on the outcome of a learning algorithm $A$, we may then know a little bit more what kind of functions will be obtained. This limits the possibilities and restricts the supremum over all the functions in $\mathcal{F}$ to the possible outcomes of the algorithm. An algorithm which always outputs the null function does not need to be studied by a uniform law of large numbers.

Let's introduce a notation for modified training sets: if $S$ denotes the initial training set, $S = \{z_1, \ldots, z_{i-1}, z_i, z_{i+1}, \ldots, z_m\}$, then $S^i$ denotes the training set after $z_i$ has been replaced by a different training example $z_i'$, that is $S^i = \{z_1, \ldots, z_{i-1}, z_i', z_{i+1}, \ldots, z_m\}$. Now, we define a notion of stability for regression.

**Definition 1 (Uniform stability)** *Let $S = \{z_1, \ldots, z_m\}$ be a training set, $S^i = S \backslash z_i$ be the training set where instance $i$ has been removed and $A$ a symmetric algorithm. We say that $A$ is $\beta$-stable if the following holds:*

$$\forall S \in \mathcal{Z}^m, \ \forall z_i', z \in \mathcal{Z}, \ |c(f_S, z) - c(f_{S^i}, z)| \leq \beta \tag{3}$$

This condition expresses that for any possible training set $S$ and any replacement example $z_i'$, the difference in cost (measured on any instance in $\mathcal{Z}$) incurred by the learning algorithm when training on $S$ and on $S^i$ is smaller than some constant $\beta$.

## 3    Main result

A stable algorithm, *i.e.* $\beta$-stable with a small $\beta$, has the property that replacing one element in its learning set does not change much its outcome. As a consequence, the empirical error, if thought as a random variable, should have a small variance. Stable algorithms can then be good candidates for their empirical error to be close to their generalization error. This assertion is formulated in the following theorem:

**Theorem 2** *Let $A$ be a $\beta$-stable algorithm, such that $0 \leq c(f_S, z) \leq M$, for all $z \in \mathcal{Z}$ and all learning set $S$. For all $\epsilon > 0$, for any $m \geq \frac{8M^2}{\epsilon^2}$, we have:*

$$\mathbf{P}_{S \sim D^m}\left[|R_m(f_S) - R(f_S)| > \epsilon\right] \leq \frac{64Mm\beta + 8M^2}{m\epsilon^2} \tag{4}$$

*and for any $m \geq 1$,*

$$\mathbf{P}_{S \sim D^m}\left[|R_m(f_S) - R(f_S)| > \epsilon + \beta\right] \leq 2\exp\left(-\frac{m\epsilon^2}{2(m\beta + M)^2}\right) \tag{5}$$

Notice that this theorem gives tight bounds when the stability $\beta$ is of the order of $1/m$. It will be proved in next section that regularization networks satisfy this requirement.

In order to prove theorem 2, one has to study the random variable $X = R(f_S) - R_m(f_S)$, which can be done using two different approaches. The first one (corresponding to the exponential inequality) uses a classical martingale inequality and is

detailed below. The second one is a bit more technical and requires to use standard proof techniques such as symmetrization. Here we only briefly sketch this proof and refer the reader to [5] for more details.

**Proof of inequality (5)** We use the following theorem :

**Theorem 3** *(McDiarmid [9]). Let $Y_1, \ldots, Y_n$ be $n$ i.i.d. random variables taking values in a set $A$, and assume that $F : A^n \to \mathbb{R}$ satisfies for $1 \le i \le n$:*

$$\sup_{y_1, \ldots, y_n, y_i' \in A} |F(y_1, \ldots, y_n) - F(y_1, \ldots, y_{i-1}, y_i', y_{i+1}, \ldots, y_n)| \le c_i$$

*then*

$$\mathbf{P}\left[|F(Y_1, \ldots, Y_n) - \mathbf{E}[F(Y_1, \ldots, Y_n)]| > \epsilon\right] \le 2e^{-2\epsilon^2 / \sum_{i=1}^n c_i^2}$$

In order to apply theorem 3, we have to bound the expectation of $X$. We begin with a useful lemma:

**Lemma 1** *For any symmetric learning algorithm we have for all $1 \le i \le m$:*

$$\mathbf{E}_{S \sim D^m}[R(f_S) - R_m(f_S)] = \mathbf{E}_{S, z_i' \sim D^{m+1}}[c(f_S, z_i') - c(f_{S^i}, z_i')]$$

**Proof:** Notice that

$$\mathbf{E}_{S \sim D^m}[R_m(f_S)] = \frac{1}{m} \sum_{j=1}^m \mathbf{E}_{S \sim D^m}[c(f_S, z_j)] = \mathbf{E}_{S \sim D^m}[c(f_S, z_i)], \forall i \in \{1, \ldots, m\}$$

by symmetry and the i.i.d. assumption. Now, by simple renaming of $z_i$ as $z_i'$ we get

$$\mathbf{E}_{S \sim D^m}[R_m(f_S)] = \mathbf{E}_{S^i \sim D^m}[c(f_{S^i}, z_i')] = \mathbf{E}_{S, z_i' \sim D^{m+1}}[c(f_{S^i}, z_i')]$$

which, with the observation that

$$\mathbf{E}_{S \sim D^m}[R(f_S)] = \mathbf{E}_{S, z_i' \sim D^{m+1}}[c(f_S, z_i')]$$

concludes the proof. ◇

Using the above lemma and the fact that $A$ is $\beta$-stable, we easily get:

$$\mathbf{E}_{S \sim D^m}[R(f_S) - R_m(f_S)] \le \mathbf{E}_{S, z_i' \sim D^{m+1}}[\beta] = \beta$$

We now have to compute the constants $c_i$ appearing in theorem 3.

We have

$$|R(f_S) - R(f_{S^i})| \le \mathbf{E}_{z \sim D}[|c(f_S, z) - c(f_{S^i}, z)|] \le \beta$$

and

$$|R_m(f_S) - R_m(f_{S^i})| \le \frac{1}{m} \sum_{j \ne i} |c(f_S, z_j) - c(f_{S^i}, z_j)| + \frac{1}{m} |c(f_S, z_i) - c(f_{S^i}, z_i')|$$

$$\le \frac{2M}{m} + \beta$$

Theorem 3 applied to $R(f_S) - R_m(f_S)$ then gives inequality (5).

**Sketch of the proof of inequality (4)** Recall Chebyshev's inequality :

$$P(|X| \ge \epsilon) \le \frac{E[X^2]}{\epsilon^2}, \tag{6}$$

for any random variable $X$. In order to apply this inequality, we have to bound $E[X^2]$. This can be done with a similar reasoning as for the expectation. Calculations are however more complex and we do not describe them here. For more details, see [5]. The result is the following:

$$E_\sigma[X^2] \le M^2/m + 8M\beta$$

which with (6) gives inequality (4) and concludes the proof.

# 4 Stability of Regularization Networks

## 4.1 Definitions

Regularization networks have been introduced in machine learning by Poggio and Girosi [10]. The relationship between these networks and the Support Vector Machines, as well as their Bayesian interpretation, make them very attractive. We consider a training set $S = \{(x_1, y_1), \dots, (x_m, y_m)\}$ with $x_i \in \mathbb{R}^d$ and $y_i \in \mathbb{R}$, that is we are in the regression setting. The regularization network technique consists in finding a function $f : \mathbb{R}^d \to \mathbb{R}$ in a space $H$ which minimizes the following functional:

$$C(f) = \frac{1}{m} \sum_{j=1}^{m} (f(x_j) - y_j)^2 + \lambda \|f\|_H^2 \qquad (7)$$

where $\|f\|_H^2$ denotes the norm in the space $H$. In this framework, $H$ is chosen to be a reproducing kernel Hilbert space (rkhs), which is basically a functional space endowed with a dot product[1]. A rkhs is defined by a *kernel* function, that is a symmetric function $k : \mathbb{R}^d \times \mathbb{R}^d \to \mathbb{R}$ which we will assume to be bounded by a constant $\kappa$ in what follows[2]. In particular the following property will hold :

$$|f(x)| \leq \|f\|_H \|k\|_H \leq \kappa \|f\|_H \qquad (8)$$

## 4.2 Stability study

In this section, we show that regularization networks are, furthermore, stable as soon as $\lambda$ is not too small.

**Theorem 4** *For regularization networks with* $\|k\|_H \leq \kappa$ *and* $(f(x) - y)^2 \leq M$,

$$R(f_S) \leq R_m(f_S) + \frac{4M\kappa}{m\lambda} + 4M \sqrt{\left(\frac{2\kappa^2}{\lambda^2} + \frac{4\kappa}{\lambda} + 2\right) \frac{\ln(2/\delta)}{m}} \qquad (9)$$

*and*

$$R(f_S) \leq R_m(f_S) + 2M \sqrt{\left(\frac{64\kappa}{\lambda} + 2\right) \frac{1}{m\delta}} \qquad (10)$$

**Proof:** Let's denote by $f_S$ the minimizer of $C$. Let's define

$$C^i(f) = \frac{1}{m} \sum_{j \neq i}^{m} (f(x_j) - y_j)^2 + \frac{1}{m}(f(x_i') - y_i')^2 + \lambda \|f\|_H^2$$

Let $f_{S^i}$ be the minimizer of $C^i$ and let $g$ denote the difference $f_{S^i} - f_S$. By simple algebra, we have for $t \in [0, 1]$

$$C(f_S) - C(f_S + tg) = -\frac{2t}{m} \sum_{j=1}^{m} (f_S(x_j) - y_j)g(x_j) - 2t\lambda <f_S, g> + t^2 A(g)$$

where $A(g)$ which is not explicitly written here is the factor of $t^2$. Similarly we have

$$
\begin{aligned}
C^i(f_{S^i}) - C^i(f_{S^i} - tg) = &\; \frac{2t}{m}\sum_{j \neq i}(f_{S^i}(x_j) - y_j)g(x_j) \\
&+ \frac{2t}{m}(f_{S^i}(x_i') - y_i')g(x_i') + 2t\lambda < f_{S^i}, g > + t^2 A^i(g)
\end{aligned}
$$

By optimality, we have

$$
C(f_S) - C(f_S + tg) \leq 0 \text{ and } C^i(f_{S^i}) - C^i(f_{S^i} - tg) \leq 0
$$

thus, summing those inequalities, dividing by $t/m$ and making $t \to 0$, we get

$$
2\sum_{j \neq i} g(x_j)^2 - 2(f_S(x_i) - y_i)g(x_i) + 2(f_{S^i}(x_i') - y_i')g(x_i') + 2m\lambda\|g\|_H^2 \leq 0
$$

which gives

$$
m\lambda\|g\|_H^2 \leq (f_S(x_i) - y_i)g(x_i) - (f_{S^i}(x_i') - y_i')g(x_i') \leq 2\sqrt{M}\kappa\|g\|_H
$$

using (8). We thus obtain

$$
\|f_{S^i} - f_S\|_H \leq 2\sqrt{M}\kappa/(m\lambda) \tag{11}
$$

and also

$$
\forall x, y \; |(f_S(x) - y)^2 - (f_{S^i}(x) - y)^2| \leq 2\sqrt{M}|f_S(x) - f_{S^i}(x)| \leq 4M\kappa/(m\lambda)
$$

We thus proved that the minimization of $C[f]$ is a $\frac{4M\kappa}{m\lambda}$-stable procedure which allows to apply theorem 2. ◇.

## 4.3 Discussion

These inequalities are both of interest since the range where they are tight is different. Indeed, (10) has a poor dependence on $\delta$ which makes it deteriorate when high confidence is sought for. However, (9) can give high confidence bounds but will be looser when $\lambda$ is small.

Moreover, results exposed by Evgeniou *et al.* [6] indicate that the optimal dependence of $\lambda$ with $m$ is obtained for $\lambda m = O(\ln \ln m)$. If we plug this into the above bounds, we can notice that (9) does not converge as $m \to \infty$. It may be conjectured that the poor estimation of the variance coming from the martingale method in McDiarmid's inequality is responsible for this effect, but a finer analysis is required to fully understand this phenomenon.

One of the interests of these results is to provide a mean for choosing the $\lambda$ parameter by minimizing the right hand side of the inequality. Usually, it is determined with a validation set: some of the data is not used during learning and $\lambda$ is chosen such that the error of $f_S$ over the validation set is minimized. The drawback of this approach is to reduce the amount of data available for learning.

## 5 Conclusion and future work

We have presented a new approach to get bounds on the generalization performance of learning algorithms which makes use of specific properties of these algorithms. The bounds we obtain do not depend on the complexity of the hypothesis class but on a measure of how stable the algorithm's output is with respect to changes in the training set.

Although this work has focused on regression, we believe that it can be extended to classification, in particular by making the stability requirement less demanding (e.g. stability in average instead of uniform stability). Future work will also aim at finding other algorithms that are stable or can be appropriately modified to exhibit the stability property. At last, a promising application of this work could be the model selection problem where one has to tune parameters of the algorithms (e.g. $\lambda$ and parameters of the kernel for regularization networks). Instead of using cross-validation, one could measure how stability is influenced by the various parameters of interest and plug these measures into theorem 2 to derive bounds on the generalization error.

## Acknowledgments

We would like to thank G. Lugosi, S. Boucheron and O. Chapelle for interesting discussions on stability and concentration inequalities. Many thanks to A. Smola and to the anonymous reviewers who helped improve the readability.

## Footnotes

[1] We do not detail here the properties of such a space and refer the reader to [2, 3] for additional details.

[2] Once again we do not give full detail of the definition of appropriate kernel functions and refer the reader to [3].

# References

[1] N. Alon, S. Ben-David, N. Cesa-Bianchi, and D. Haussler. Scale-sensitive dimensions, uniform convergence and learnability. *Journal of the ACM*, 44(4):615–631, 1997.

[2] N. Aronszajn. Theory of reproducing kernels. *Trans. Amer. Math. Soc.*, 68:337–404, 1950.

[3] M. Atteia. *Hilbertian Kernels and splines functions*. Studies in computational mathematics 4. North-Holland, 1992.

[4] L.P. Devroye and T.J. Wagner. Distribution-free performance bounds for potential function rules. *IEEE Trans. on Information Theory*, 25(5):202–207, 1979.

[5] A. Elisseeff. A study about algorithmic stability and its relation to generalization performances. Technical report, Laboratoire ERIC, Univ. Lyon 2, 2000.

[6] T. Evgeniou, M. Pontil, and T. Poggio. A unified framework for regularization networks and support vector machines. Technical Memo AIM-1654, Massachusetts Institute of Technology, Artificial Intelligence Laboratory, December 1999.

[7] Y. Freund. Self bounding learning algorithms. In *Proceedings of the 11th Annual Conference on Computational Learning Theory (COLT-98)*, pages 247–258, New York, July 24–26 1998. ACM Press.

[8] M. Kearns and D. Ron. Algorithmic stability and sanity-check bounds for leave-one-out cross-validation. *Neural Computation*, 11(6):1427–1453, 1999.

[9] C. McDiarmid. On the method of bounded differences. In *Surveys in Combinatorics*, pages 148–188. Cambridge University Press, Cambridge, 1989.

[10] T. Poggio and F. Girosi. Regularization algorithms for learning that are equivalent to multilayer networks. *Science*, 247:978–982, 1990.

[11] J. Shawe-Taylor, P. L. Bartlett, R. C. Williamson, and M. Anthony. A framework for structural risk minimization. In *Proc. 9th Annu. Conf. on Comput. Learning Theory*, pages 68–76. ACM Press, New York, NY, 1996.

[12] J. Shawe-Taylor and R. C. Williamson. Generalization performance of classifiers in terms of observed covering numbers. In Paul Fischer and Hans Ulrich Simon, editors, *Proceedings of the 4th European Conference on Computational Learning Theory (Eurocolt-99)*, volume 1572 of *LNAI*, pages 274–284, Berlin, March 29–31 1999. Springer.
